# Parametric Bandits:
# The Generalized Linear Case

**Sarah Filippi**
LTCI
Telecom ParisTech et CNRS
Paris, France
filippi@telecom-paristech.fr

**Olivier Cappé**
LTCI
Telecom ParisTech et CNRS
Paris, France
cappe@telecom-paristech.fr

**Aurélien Garivier**
LTCI
Telecom ParisTech et CNRS
Paris, France
garivier@telecom-paristech.fr

**Csaba Szepesvári**
RLAI Laboratory
University of Alberta
Edmonton, Canada
szepesva@ualberta.ca

## Abstract

We consider structured multi-armed bandit problems based on the Generalized Linear Model (GLM) framework of statistics. For these bandits, we propose a new algorithm, called GLM-UCB. We derive finite time, high probability bounds on the regret of the algorithm, extending previous analyses developed for the linear bandits to the non-linear case. The analysis highlights a key difficulty in generalizing linear bandit algorithms to the non-linear case, which is solved in GLM-UCB by focusing on the reward space rather than on the parameter space. Moreover, as the actual effectiveness of current parameterized bandit algorithms is often poor in practice, we provide a tuning method based on asymptotic arguments, which leads to significantly better practical performance. We present two numerical experiments on real-world data that illustrate the potential of the GLM-UCB approach.

**Keywords**: multi-armed bandit, parametric bandits, generalized linear models, UCB, regret minimization.

## 1 Introduction

In the classical $K$-armed bandit problem, an agent selects at each time step one of the $K$ arms and receives a reward that depends on the chosen action. The aim of the agent is to choose the sequence of arms to be played so as to maximize the cumulated reward. There is a fundamental trade-off between gathering experimental data about the reward distribution (exploration) and exploiting the arm which seems to be the most promising.

In the basic multi-armed bandit problem, also called the independent bandits problem, the rewards are assumed to be random and distributed independently according to a probability distribution that is specific to each arm –see [1, 2, 3, 4] and references therein. Recently, *structured bandit problems* in which the distributions of the rewards pertaining to each arm are connected by a common unknown parameter have received much attention [5, 6, 7, 8, 9]. This model is motivated by the many practical applications where the number of arms is large, but the payoffs are interrelated. Up to know, two different models were studied in the literature along these lines. In one model, in each times step, a side-information, or context, is given to the agent first. The payoffs of the arms depend both on this side information and the index of the arm. Thus the optimal arm changes with the context [5, 6, 9]. In the second, simpler model, that we are also interested in here, there is no side-information, but the agent is given a model that describes the possible relations

between the arms' payoffs. In particular, in "linear bandits" [10, 8, 11, 12], each arm $a \in A$ is associated with some $d$-dimensional vector $m_a \in \mathbb{R}^d$ known to the agent. The expected payoffs of the arms are given by the inner product of their associated vector and some fixed, but initially unknown parameter vector $\theta_*$. Thus, the expected payoff of arm $a$ is $m'_a \theta_*$, which is linear in $\theta_*$.[1]

In this article, we study a richer *generalized linear model* (GLM) in which the expectation of the reward conditionally to the action $a$ is given by $\mu(m'_a \theta_*)$, where $\mu$ is a real-valued, non-linear function called the (inverse) link function. This generalization allows to consider a wider class of problems, and in particular cases where the rewards are counts or binary variables using, respectively, Poisson or logistic regression. Obviously, this situation is very common in the fields of marketing, social networking, web-mining (see example of Section 5.2 below) or clinical studies.

Our first contribution is an "optimistic" algorithm, termed GLM-UCB, inspired by the *Upper Confidence Bound* (UCB) approach [2]. GLM-UCB generalizes the algorithms studied by [10, 8, 12]. Our next contribution are finite-time bounds on the statistical performance of this algorithm. In particular, we show that the performance depends on the dimension of the parameter but not on the number of arms, a result that was previously known in the linear case. Interestingly, the GLM-UCB approach takes advantage of the particular structure of the parameter estimate of generalized linear models and *operates only in the reward space*. In contrast, the parameter-space confidence region approach adopted by [8, 12] appears to be harder to generalize to non-linear regression models. Our second contribution is a tuning method based on asymptotic arguments. This contribution addresses the poor empirical performance of the current algorithms that we have observed for small or moderate sample-sizes when these algorithms are tuned based on finite-sample bounds.

The paper is organized as follows. The generalized linear bandit model is presented in Section 2, together with a brief survey of needed statistical results. Section 3 is devoted to the description of the GLM-UCB algorithm, which is compared to related approaches. Section 4 presents our regret bounds, as well as a discussion, based on asymptotic arguments, on the optimal tuning of the method. Section 5 reports the results of two experiments on real data sets.

## 2 Generalized Linear Bandits, Generalized Linear Models

We consider a structured bandit model with a finite, but possibly very large, number of arms. At each time $t$, the agent chooses an arm $A_t$ from the set $A$ (we shall denote the cardinality of $A$ by $K$). The prior knowledge available to the agent consists of a collection of vectors $\{m_a\}_{a \in A}$ of features which are specific to each arm and a so-called (inverse) *link function* $\mu : \mathbb{R} \to \mathbb{R}$.

The *generalized linear bandit model* investigated in this work is based on the assumption that the payoff $R_t$ received at time $t$ is conditionally independent of the past payoffs and choices and it satisfies

$$\mathbb{E}\left[R_t | A_t\right] = \mu(m'_{A_t} \theta_*),\tag{1}$$

for some unknown parameter vector $\theta_* \in \mathbb{R}^d$. This framework generalizes the linear bandit model considered by [10, 8, 12]. Just like the linear bandit model builds on linear regression, our model capitalizes on the well-known statistical framework of Generalized Linear Models (GLMs). The advantage of this framework is that it allows to address various, specific reward structures widely found in applications. For example, when rewards are binary-valued, a suitable choice of $\mu$ is $\mu(x) = \exp(x)/(1 + \exp(x))$, leading to the *logistic regression model*. For integer valued rewards, the choice $\mu(x) = \exp(x)$ leads to the *Poisson regression model*. This can be easily extended to the case of *multinomial (or polytomic) logistic regression*, which is appropriate to model situations in which the rewards are associated with categorical variables.

To keep this article self-contained, we briefly review the main properties of GLMs [13]. A univariate probability distribution is said to belong to a *canonical exponential family* if its density with respect to a reference measure is given by

$$p_\beta(r) = \exp\left(r\beta - b(\beta) + c(r)\right),\tag{2}$$

where $\beta$ is a real parameter, $c(\cdot)$ is a real function and the function $b(\cdot)$ is assumed to be twice continuously differentiable. This family contains the Gaussian and Gamma distributions when the reference measure is the Lebesgue measure and the Poisson and Bernoulli distributions when the

reference measure is the counting measure on the integers. For a random variable $R$ with density defined in (2), $\mathbb{E}(R) = \dot{b}(\beta)$ and $\mathrm{Var}(R) = \ddot{b}(\beta)$, where $\dot{b}$ and $\ddot{b}$ denote, respectively, the first and second derivatives of $b$. In addition, $\ddot{b}(\beta)$ can also be shown to be equal to the Fisher information matrix for the parameter $\beta$. The function $b$ is thus strictly convex.

Now, assume that, in addition to the response variable $R$, we have at hand a vector of covariates $X \in \mathbb{R}^d$. The canonical GLM associated to (2) postulates that $p_\theta(r|x) = p_{x'\theta}(r)$, where $\theta \in \mathbb{R}^d$ is a vector of parameter. Denote by $\mu = \dot{b}$ the so-called *inverse link function*. From the properties of $b$, we know that $\mu$ is continuously differentiable, strictly increasing, and thus one-to-one. The maximum likelihood estimator $\hat{\theta}_t$, based on observations $(R_1, X_1), \ldots (R_{t-1}, X_{t-1})$, is defined as the maximizer of the function

$$\sum_{k=1}^{t-1} \log p_\theta(R_k|X_k) = \sum_{k=1}^{t-1} R_k X'_k \theta - b(X'_k \theta) + c(R_k) \,,$$

a strictly concave function in $\theta$.[2] Upon differentiating, we obtain that $\hat{\theta}_t$ is the unique solution of the following estimating equation

$$\sum_{k=1}^{t-1} \left( R_k - \mu(X'_k \theta) \right) X_k = 0 \,, \tag{3}$$

where we have used the fact that $\mu = \dot{b}$. In practice, the solution of (3) may be found efficiently using, for instance, Newton's algorithm.

A semi-parametric version of the above model is obtained by assuming only that $\mathbb{E}_\theta[R|X] = \mu(X'\theta)$ without (much) further assumptions on the conditional distribution of $R$ given $X$. In this case, the estimator obtained by solving (3) is referred to as the *maximum quasi-likelihood estimator*. It is a remarkable fact that this estimator is consistent under very general assumptions as long as the design matrix $\sum_{k=1}^{t-1} X_k X'_k$ tends to infinity [14]. As we will see, this matrix also plays a crucial role in the algorithm that we propose for bandit optimization in the generalized linear bandit model.

## 3 The GLM-UCB Algorithm

According to (1), the agent receives, upon playing arm $a$, a random reward whose expected value is $\mu(m'_a \theta_*)$, where $\theta_* \in \Theta$ is the unknown parameter. The parameter set $\Theta$ is an arbitrary closed subset of $\mathbb{R}^d$. Any arm with largest expected reward is called *optimal*. The aim of the agent is to quickly find an optimal arm in order to maximize the received rewards. The *greedy action* $\mathrm{argmax}_{a \in \mathsf{A}} \mu(m'_a \hat{\theta}_t)$ may lead to an unreliable algorithm which does not sufficiently explore to guarantee the selection of an optimal arm. This issue can be addressed by resorting to an "optimistic approach". As described by [8, 12] in the linear case, an optimistic algorithm consists in selecting, at time $t$, the arm

$$A_t = \operatorname*{argmax}_a \max_\theta \mathbb{E}_\theta \left[ R_t \mid A_t = a \right] \text{ s.t. } \|\theta - \hat{\theta}_t\|_{M_t} \le \rho(t) \,, \tag{4}$$

where $\rho$ is an appropriate, "slowly increasing" function,

$$M_t = \sum_{k=1}^{t-1} m_{A_k} m'_{A_k} \tag{5}$$

is the design matrix corresponding to the first $t-1$ timesteps and $\|v\|_M = \sqrt{v'Mv}$ denotes the matrix norm induced by the positive semidefinite matrix $M$. The region $\|\theta - \hat{\theta}_t\|_{M_t} \le \rho(t)$ is a confidence ellipsoid around the estimated parameter $\hat{\theta}_t$. Generalizing this approach beyond the case of linear link functions looks challenging. In particular, in GLMs, the relevant confidence regions may have a more complicated geometry in the parameter space than simple ellipsoids. As a consequence, the benefits of this form of optimistic algorithms appears dubious.[3]

An alternative approach consists in directly determining an upper confidence bound for the expected reward of each arm, thus choosing the action $a$ that maximizes

$$\mathbb{E}_{\hat{\theta}_t}\left[R_t \,|\, A_t = a\right] + \rho(t)\|m_a\|_{M_t^{-1}}.$$

In the linear case the two approaches lead to the same solution [12]. Interestingly, for non-linear bandits, the second approach looks more appropriate.

In the rest of this section, we apply this second approach to the GLM bandit model defined in (1). According to (3), the maximum quasi-likelihood estimator of the parameter in the GLM is the unique solution of the estimating equation

$$\sum_{k=1}^{t-1}\left(R_k - \mu(m'_{A_k}\hat{\theta}_t)\right)m_{A_k} = 0\,, \tag{6}$$

where $A_1,\dots,A_{t-1}$ denote the arms played so far and $R_1,\dots,R_{t-1}$ are the corresponding rewards. Let $g_t(\theta) = \sum_{k=1}^{t-1}\mu(m'_{A_k}\theta)m_{A_k}$ be the invertible function such that the estimated parameter $\hat{\theta}_t$ satisfies $g_t(\hat{\theta}_t) = \sum_{k=1}^{t-1}R_k m_{A_k}$. Since $\hat{\theta}_t$ might be outside of the set of admissible parameters $\Theta$, we "project it" to $\Theta$, to obtain $\tilde{\theta}_t$:

$$\tilde{\theta}_t = \operatorname*{argmin}_{\theta\in\Theta}\left\|g_t(\theta) - g_t(\hat{\theta}_t)\right\|_{M_t^{-1}} = \operatorname*{argmin}_{\theta\in\Theta}\left\|g_t(\theta) - \sum_{k=1}^{t-1}R_k m_{A_k}\right\|_{M_t^{-1}}. \tag{7}$$

Note that if $\hat{\theta}_t \in \Theta$ (which is easy to check and which happened to hold always in the examples we dealt with) then we can let $\tilde{\theta}_t = \hat{\theta}_t$. This is important since computing $\tilde{\theta}_t$ is non-trivial and we can save this computation by this simple check. The proposed algorithm, GLM-UCB, is as follows:

---
**Algorithm 1** GLM-UCB
---
1: **Input:** $\{m_a\}_{a\in\mathsf{A}}$
2: Play actions $a_1,\dots,a_d$, receive $R_1,\dots,R_d$.
3: **for** $t > d$ **do**
4:    Estimate $\hat{\theta}_t$ according to (6)
5:    **if** $\hat{\theta}_t \in \Theta$ **let** $\tilde{\theta}_t = \hat{\theta}_t$ **else** compute $\tilde{\theta}_t$ according to (7)
6:    Play the action $A_t = \operatorname{argmax}_a\left\{\mu(m'_a\tilde{\theta}_t) + \rho(t)\|m_a\|_{M_t^{-1}}\right\}$, receive $R_t$
7: **end for**

---

At time $t$, for each arm $a$, an upper bound $\mu(m'_a\tilde{\theta}_t) + \beta_t^a$ is computed, where the "exploration bonus" $\beta_t^a = \rho(t)\|m_a\|_{M_t^{-1}}$ is the product of two terms. The quantity $\rho(t)$ is a slowly increasing function; we prove in Section 4 that $\rho(t)$ can be set to guarantee high-probability bounds on the expected regret (for the actual form used, see (8)). Note that the leading term of $\beta_t^a$ is $\|m_a\|_{M_t^{-1}}$ which decreases to zero as $t$ increases.

As we are mostly interested in the case when the number of arms $K$ is much larger than the dimension $d$, the algorithm is simply initialized by playing actions $a_1,\dots,a_d$ such that the vectors $m_{a_1}\dots,m_{a_d}$ form a basis of $\mathcal{M} = \operatorname{span}(m_a, a \in \mathsf{A})$. Without loss of generality, here and in what follows we assume that the dimension of $\mathcal{M}$ is equal to $d$. Then, by playing $a_1,\dots,a_d$ in the first $d$ steps the agent ensures that $M_t$ is invertible for all $t$. An alternative strategy would be to initialize $M_0 = \lambda_0 I$, where $I$ is the $d \times d$ identify matrix.

### 3.1 Discussion

The purpose of this section is to discuss some properties of Algorithm 1, and in particular the interpretation of the role played by $\|m_a\|_{M_t^{-1}}$.

**Generalizing UCB** The standard UCB algorithm for $K$ arms [2] can be seen as a special case of GLM-UCB where the vectors of covariates associated with the arms form an orthogonal system and $\mu(x) = x$. Indeed, take $d = K$, $\mathsf{A} = \{1,\dots,K\}$, define the vectors $\{m_a\}_{a\in\mathsf{A}}$ as the canonical basis $\{\mathbf{e}_a\}_{a\in\mathsf{A}}$ of $\mathbb{R}^d$, and take $\theta \in \mathbb{R}^d$ the vector whose component $\theta_a$ is the expected reward for arm $a$.

Then, $M_t$ is a diagonal matrix whose $a$-th diagonal element is the number $N_t(a)$ of times the $a$-th arm has been played up to time $t$. Therefore, the exploration bonus in GLM-UCB is given by $\beta_t^a = \rho(t)/\sqrt{N_t(a)}$. Moreover, the maximum quasi-likelihood estimator $\hat{\theta}_t$ satisfies $\bar{R}_t^a = \hat{\theta}_t(a)$ for all $a \in \mathsf{A}$, where $\bar{R}_t^a = \frac{1}{N_t(a)}\sum_{k=1}^{t-1} \mathbb{I}_{\{A_t=a\}} R_k$ is the empirical mean of the rewards received while playing arm $a$. Algorithm 1 then reduces to the familiar UCB algorithm. In this case, it is known that the expected cumulated regret can be controlled upon setting the slowly varying function $\rho$ to $\rho(t) = \sqrt{2\log(t)}$, assuming that the range of the rewards is bounded by one [2].

**Generalizing linear bandits** Obviously, setting $\mu(x) = x$, we obtain a linear bandit model. In this case, assuming that $\Theta = \mathbb{R}^d$, the algorithm will reduce to those described in the papers [8, 12]. In particular, the maximum quasi-likelihood estimator becomes the least-squares estimator and as noted earlier, the algorithm behaves identically to one which chooses the parameter optimistically within the confidence ellipsoid $\{\theta : \|\theta - \hat{\theta}_t\|_{M_t} \le \rho(t)\}$.

**Dependence in the Number of Arms** In contrast to an algorithm such as UCB, Algorithm 1 does not need that all arms be played even once.[4] To understand this phenomenon, observe that, as $M_{t+1} = M_t + m_{A_t}m'_{A_t}$, $\|m_a\|^2_{M_{t+1}^{-1}} = \|m_a\|^2_{M_t^{-1}} - \left(m'_a M_t^{-1} m_{A_t}\right)^2/(1 + \|m_{A_t}\|^2_{M_t^{-1}})$ for any arm $a$. Thus the exploration bonus $\beta_{t+1}^a$ decreases for all arms, except those which are exactly orthogonal to $m_{A_t}$ (in the $M_t^{-1}$ metric). The decrease is most significant for arms that are colinear to $m_{A_t}$. This explains why the regret bounds obtained in Theorems 1 and 2 below depend on $d$ but not on $K$.

## 4 Theoretical analysis

In this section we first give our finite sample regret bounds and then show how the algorithm can be tuned based on asymptotic arguments.

### 4.1 Regret Bounds

To quantify the performance of the GLM-UCB algorithm, we consider the *cumulated (pseudo) regret* defined as the expected difference between the optimal reward obtained by always playing an optimal arm and the reward received following the algorithm:

$$\text{Regret}_T = \sum_{t=1}^{T} \mu(m'_{a^*}\theta_*) - \mu(m'_{A_t}\theta_*)\,.$$

For the sake of the analysis, in this section we shall assume that the following assumptions hold:

**Assumption 1.** *The link function* $\mu : \mathbb{R} \to \mathbb{R}$ *is continuously differentiable, Lipschitz with constant* $k_\mu$ *and such that* $c_\mu = \inf_{\theta \in \Theta, a \in \mathsf{A}} \dot{\mu}(m'_a\theta) > 0$.

For the logistic function $k_\mu = 1/4$, while the value of $c_\mu$ depends on $\sup_{\theta \in \Theta, a \in \mathsf{A}} |m'_a\theta|$.

**Assumption 2.** *The norm of covariates in* $\{m_a : a \in \mathsf{A}\}$ *is bounded: there exists* $c_m < \infty$ *such that for all* $a \in \mathsf{A}$, $\|m_a\|_2 \le c_m$.

Finally, we make the following assumption on the rewards:

**Assumption 3.** *There exists* $R_{\max} > 0$ *such that for any* $t \ge 1$, $0 \le R_t \le R_{\max}$ *holds a.s. Let* $\epsilon_t = R_t - \mu(m'_{A_t}\theta_*)$. *For all* $t \ge 1$, *it holds that* $\mathbb{E}\left[\epsilon_t | m_{A_t}, \epsilon_{t-1}, \ldots, m_{A_2}, \epsilon_1, m_{A_1}\right] = 0$ *a.s.*

As for the standard UCB algorithm, the regret can be analyzed in terms of the difference between the expected reward received playing an optimal arm and that of the best sub-optimal arm:

$$\Delta(\theta_*) = \min_{a:\mu(m'_a\theta_*)<\mu(m'_{a^*}\theta_*)} \mu(m'_{a^*}\theta_*) - \mu(m'_a\theta_*)\,.$$

Theorem 1 establishes a high probability bound on the regret underlying using GLM-UCB with

$$\rho(t) = \frac{2k_\mu \kappa R_{\max}}{c_\mu}\sqrt{2d\log(t)\log(2\,d\,T/\delta)}\,, \tag{8}$$

where $T$ is the fixed time horizon, $\kappa = \sqrt{3 + 2\log(1 + 2c_m^2/\lambda_0)}$ and $\lambda_0$ denotes the smallest eigenvalue of $\sum_{i=1}^{d} m_{a_i} m'_{a_i}$, which by our previous assumption is positive.

**Theorem 1** (Problem Dependent Upper Bound). *Let $s = \max(1, c_m^2/\lambda_0)$. Then, under Assumptions 1–3, for all $T \geq 1$, the regret satisfies:*

$$\mathbb{P}\left( Regret_T \leq (d+1)R_{\max} + \frac{C\,d^2}{\Delta(\theta_*)} \log^2\left[s\,T\right] \log\left[\frac{2dT}{\delta}\right] \right) \geq 1 - \delta \quad with \quad C = \frac{32\kappa^2 R_{\max}^2 k_\mu^2}{c_\mu^2}.$$

Note that the above regret bound depends on the true value of $\theta_*$ through $\Delta(\theta_*)$. The following theorem provides an upper-bound of the regret independently of the $\theta_*$.

**Theorem 2** (Problem Independent Upper Bound). *Let $s = \max(1, c_m^2/\lambda_0)$. Then, under Assumptions 1–3, for all $T \geq 1$, the regret satisfies*

$$\mathbb{P}\left( Regret_T \leq (d+1)R_{\max} + Cd\log\left[s\,T\right] \sqrt{T\log\left[\frac{2dT}{\delta}\right]} \right) \geq 1 - \delta \quad with \quad C = \frac{8R_{\max}k_\mu\kappa}{c_\mu}.$$

The proofs of Theorems 1–2 can be found in the supplementary material. The main idea is to use the explicit form of the estimator given by (6) to show that

$$\left| \mu(m'_{A_t}\theta_*) - \mu(m'_{A_t}\hat{\theta}_t) \right| \leq \frac{k_\mu}{c_\mu} \|m_{A_t}\|_{M_t^{-1}} \left\| \sum_{k=1}^{t-1} m_{A_k}\,\epsilon_k \right\|_{M_t^{-1}}.$$

Bounding the last term on the right-hand side is then carried out following the lines of [12].

## 4.2 Asymptotic Upper Confidence Bound

Preliminary experiments carried out using the value of $\rho(t)$ defined equation (8), including the case where $\mu$ is the identity function –i.e., using the algorithm described by [8, 12], revealed poor performance for moderate sample sizes. A look into the proof of the regret bound easily explains this observation as the mathematical involvement of the arguments is such that some approximations seem unavoidable, in particular several applications of the Cauchy-Schwarz inequality, leading to pessimistic confidence bounds. We provide here some asymptotic arguments that suggest to choose significantly smaller exploration bonuses, which will in turn be validated by the numerical experiments presented in Section 5.

Consider the canonical GLM associated with an inverse link function $\mu$ and assume that the vectors of covariates $X$ are drawn independently under a fixed distribution. This *random design* model would for instance describe the situation when the arms are drawn randomly from a fixed distribution. Standard statistical arguments show that the Fisher information matrix pertaining to this model is given by $J = \mathbb{E}[\dot{\mu}(X'\theta_*)XX']$ and that the maximum likelihood estimate $\hat{\theta}_t$ is such that $t^{-1/2}(\hat{\theta}_t - \theta_*) \xrightarrow{\mathcal{D}} \mathcal{N}(0, J^{-1})$, where $\xrightarrow{\mathcal{D}}$ stands for convergence in distribution. Moreover, $t^{-1}M_t \xrightarrow{a.s.} \Sigma$ where $\Sigma = \mathbb{E}[XX']$. Hence, using the delta-method and Slutsky's lemma

$$\|m_a\|_{M_t^{-1}}^{-1}(\mu(m'_a\hat{\theta}_t) - \mu(m'_a\theta_*)) \xrightarrow{\mathcal{D}} \mathcal{N}(0, \dot{\mu}(m'_a\theta_*)\|m'_a\|_{\Sigma^{-1}}^{-2}\|m'_a\|_{J^{-1}}^{2}).$$

The right-hand variance is smaller than $k_\mu/c_\mu$ as $J \succeq c_\mu\Sigma$. Hence, for *any sampling distribution* such that $J$ and $\Sigma$ are positive definite and sufficiently large $t$ and small $\delta$,

$$\mathbb{P}\left( \|m_a\|_{M_t^{-1}}^{-1}(\mu(m'_a\hat{\theta}_t) - \mu(m'_a\theta_*)) > \sqrt{2k_\mu/c_\mu \log(1/\delta)} \right)$$

is asymptotically bounded by $\delta$. Based on the above asymptotic argument, we postulate that using $\rho(t) = \sqrt{2k_\mu/c_\mu \log(t)}$, i.e., inflating the exploration bonus by a factor of $\sqrt{k_\mu/c_\mu}$ compared to the usual UCB setting, is sufficient. This is the setting used in the simulations below.

## 5 Experiments

To the best of our knowledge, there is currently no public benchmark available to test bandit methods on real world data. On simulated data, the proposed method unsurprisingly outperforms its competitors when the data is indeed simulated from a well-specified generalized linear model. In order to evaluate the potential of the method in more challenging scenarios, we thus carried out two experiments using real world datasets.

## 5.1 Forest Cover Type Data

In this first experiment, we test the performance of the proposed method on a toy problem using the "Forest Cover Type dataset" from the UCI repository. The dataset (centered and normalized with constant covariate added, resulting in 11-dimensional vectors, ignoring all categorical variables) has been partitioned into $K = 32$ clusters using unsupervised k-means. The values of the response variable for the data points assigned to each cluster are viewed as the outcomes of an arm while the centroid of the cluster is taken as the 11-dimensional vector of covariates characteristic of the arm. To cast the problem into the logistic regression framework, each response variable is binarized by associating the first class ("Spruce/Fir") to a response $R = 1$ and all other six classes to $R = 0$. The proportions of responses equal to 1 in each cluster (or, in other word, the expected reward associated with each arm) ranges from 0.354 to 0.992, while the proportion on the complete set of 581,012 data points is equal to 0.367. In effect, we try to locate as fast as possible the cluster that contains the maximal proportion of trees from a given species. We are faced with a 32-arm problem in a 11-dimensional space with binary rewards. Obviously, the logistic regression model is not satisfied, although we do expect some regularity with respect to the position of the cluster's centroid as the logistic regression trained on all data reaches a 0.293 misclassification rate.

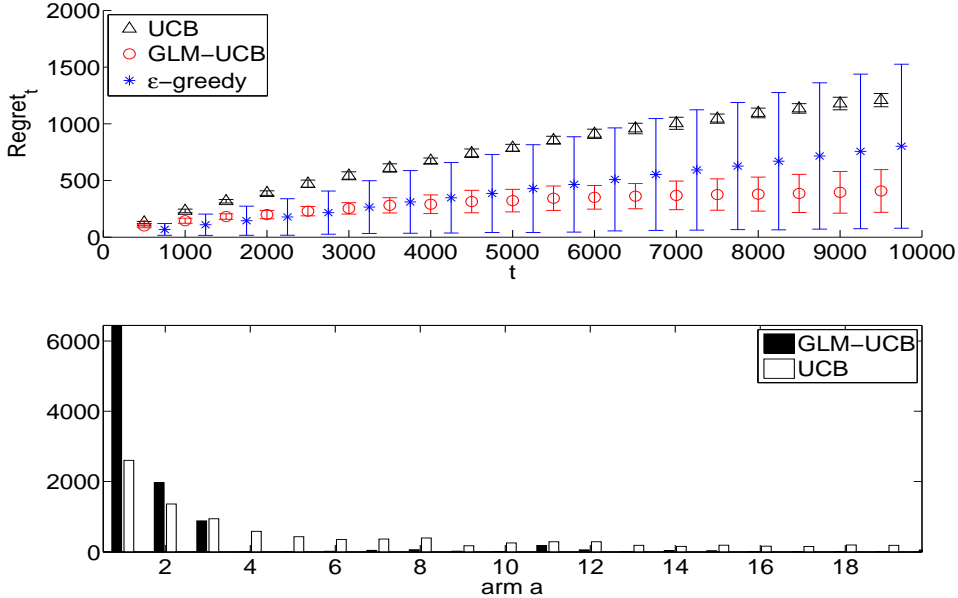

Figure 1: Top: Regret of the UCB, GLM-UCB and the $\epsilon$-greedy algorithms. Bottom: Frequencies of the 20 best arms draws using the UCB and GLM-UCB.

We compare the performance of three algorithms. First, the GLM-UCB algorithm, with parameters tuned as indicated in Section 4.2. Second, the standard UCB algorithm that ignores the covariates. Third, an $\epsilon$-greedy algorithm that performs logistic regression and plays the best estimated action, $A_t = \mathrm{argmax}_a \, \mu(m'_a \hat{\theta}_t)$, with probability $1 - \epsilon$ (with $\epsilon = 0.1$). We observe in the top graph of Figure 1 that the GLM-UCB algorithm achieves the smallest average regret by a large margin. When the parameter is well estimated, the greedy algorithm may find the best arm in little time and then leads to small regrets. However, the exploration/exploitation tradeoff is not correctly handled by the $\epsilon$-greedy approach causing a large variability in the regret. The lower plot of Figure 1 shows the number of times each of the 20 best arms have been played by the UCB and GLM-UCB algorithms. The arms are sorted in decreasing order of expected reward. It can be observed that GML-UCB only plays a small subset of all possible arms, concentrating on the bests. This behavior is made possible by the predictive power of the covariates: by sharing information between arms, it is possible to obtain sufficiently accurate predictions of the expected rewards of all actions, even for those that have never (or rarely) been played.

## 5.2 Internet Advertisement Data

In this experiment, we used a large record of the activity of internet users provided by a major ISP. The original dataset logs the visits to a set of 1222 pages over a six days period corresponding to about $5.10^8$ page visits. The dataset also contains a record of the users clicks on the ads that were presented on these pages. We worked with a subset of 208 ads and $3.10^5$ users. The pages (ads) were partitioned in 10 (respectively, 8) categories using Latent Dirichlet Allocation [15] applied to their respective textual content (in the case of ads, the textual content was that of the page pointed to by the ad's link). This second experiment is much more challenging, as the predictive power of the sole textual information turns out to be quite limited (for instance, Poisson regression trained on the entire data does not even correctly identify the best arm).

The action space is composed of the 80 pairs of pages and ads categories: when a pair is chosen, it is presented to a group of 50 users, randomly selected from the database, and the reward is the number of recorded clicks. As the average reward is typically equal to 0.15, we use a logarithmic link function corresponding to Poisson regression. The vector of covariates for each pair is of dimension 19: it is composed of an intercept followed by the concatenation of two vectors of dimension 10 and 8 representing, respectively, the categories of the pages and the ads. In this problem, the covariate vectors do not span the entire space; to address this issue, it is sufficient to consider the pseudo-inverse of $M_t$ instead of the inverse.

On this data, we compared the GLM-UCB algorithm with the two alternatives described in Section 5.1. Figure 2 shows that GLM-UCB once again outperforms its competitors, even though the margin over UCB is now less remarkable. Given the rather limited predictive power of the covariates in this example, this is an encouraging illustration of the potential of techniques which use vectors of covariates in real-life applications.

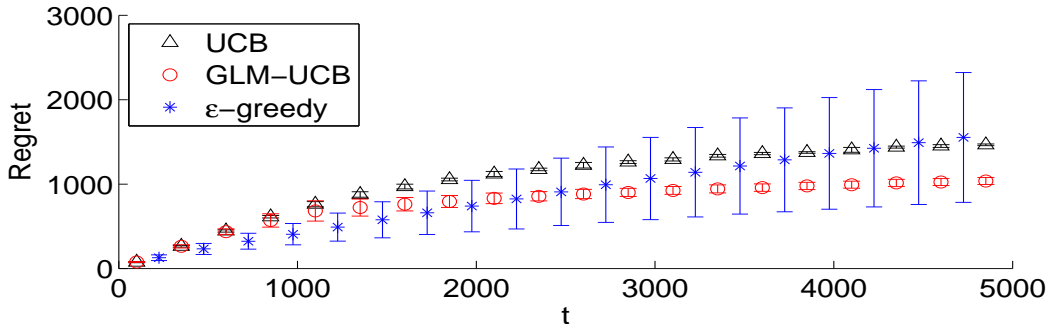

Figure 2: Comparison of the regret of the UCB, GLM-UCB and the $\epsilon$-greedy ($\epsilon = 0.1$) algorithm on the advertisement dataset.

## 6 Conclusions

We have introduced an approach that generalizes the linear regression model studied by [10, 8, 12]. As in the original UCB algorithm, the proposed GLM-UCB method operates directly in the reward space. We discussed how to tune the parameters of the algorithm to avoid exaggerated optimism, which would slow down learning. In the numerical simulations, the proposed algorithm was shown to be competitive and sufficiently robust to tackle real-world problems. An interesting open problem (already challenging in the linear case) consists in tightening the theoretical results obtained so far in order to bridge the gap between the existing (pessimistic) confidence bounds and those suggested by the asymptotic arguments presented in Section 4.2, which have been shown to perform satisfactorily in practice.

### Acknowledgments

This work was supported in part by AICML, AITF, NSERC, PASCAL2 under nº216886, the DARPA GALE project under nºHR0011-08-C-0110 and Orange Labs under contract nº289365.

## Footnotes

[1]Throughout the paper we use the prime to denote transposition.

[2] Here, and in what follows log denotes the natural logarithm.

[3] Note that maximizing $\mu(m'_a \theta)$ over a convex confidence region is equivalent to maximizing $m'_a \theta$ over the same region since $\mu$ is strictly increasing. Thus, computationally, this approach is not more difficult than it is for the linear case.

[4]Of course, the linear bandit algorithms also share this property with our algorithm.

## References

[1] T.L. Lai and H. Robbins. Asymptotically efficient adaptive allocation rules. *Advances in Applied Mathematics*, 6(1):4–22, 1985.

[2] P. Auer, N. Cesa-Bianchi, and P. Fischer. Finite-time analysis of the multiarmed bandit problem. *Machine Learning*, 47(2):235–256, 2002.

[3] N. Cesa-Bianchi and G. Lugosi. *Prediction, learning, and games*. Cambridge Univ Pr, 2006.

[4] J. Audibert, R. Munos, and Cs. Szepesvári. Tuning bandit algorithms in stochastic environments. *Lecture Notes in Computer Science*, 4754:150, 2007.

[5] C.C. Wang, S.R. Kulkarni, and H.V. Poor. Bandit problems with side observations. *IEEE Transactions on Automatic Control*, 50(3):338–355, 2005.

[6] J. Langford and T. Zhang. The epoch-greedy algorithm for multi-armed bandits with side information. *Advances in Neural Information Processing Systems*, pages 817–824, 2008.

[7] S. Pandey, D. Chakrabarti, and D. Agarwal. Multi-armed bandit problems with dependent arms. *International Conference on Machine learning*, pages 721–728, 2007.

[8] V. Dani, T.P. Hayes, and S.M. Kakade. Stochastic linear optimization under bandit feedback. *Conference on Learning Theory*, 2008.

[9] S.M. Kakade, S. Shalev-Shwartz, and A. Tewari. Efficient bandit algorithms for online multiclass prediction. In *Proceedings of the 25th International Conference on Machine learning*, pages 440–447. ACM, 2008.

[10] P. Auer. Using confidence bounds for exploitation-exploration trade-offs. *Journal of Machine Learning Research*, 3:397–422, 2002.

[11] Y. Abbasi-Yadkori, A. Antos, and Cs. Szepesvári. Forced-exploration based algorithms for playing in stochastic linear bandits. In *COLT Workshop on On-line Learning with Limited Feedback*, 2009.

[12] P. Rusmevichientong and J.N. Tsitsiklis. Linearly parameterized bandits. *Mathematics of Operations Research*, 35(2):395–411, 2010.

[13] P. McCullagh and J. A. Nelder. *Generalized Linear Models*. Chapman and Hall, 1989.

[14] K. Chen, I. Hu, and Z. Ying. Strong consistency of maximum quasi-likelihood estimators in generalized linear models with fixed and adaptive designs. *Annals of Statistics*, 27(4):1155–1163, 1999.

[15] David M. Blei, Andrew Y. Ng, and Michael I. Jordan. Latent Dirichlet allocation. *Advances in Neural Information Processing Systems*, 14:601–608, 2002.

[16] V.H. De La Pena, M.J. Klass, and T.L. Lai. Self-normalized processes: exponential inequalities, moment bounds and iterated logarithm laws. *Annals of Probability*, 32(3):1902–1933, 2004.

[17] P. Rusmevichientong and J.N. Tsitsiklis. Linearly parameterized bandits. Arxiv preprint arXiv:0812.3465v2, 2008.

